# Computing with Arrays of Bell-Shaped and Sigmoid Functions

**Pierre Baldi***
Jet Propulsion Laboratory
California Institute of Technology
Pasadena, CA 91109

## Abstract

We consider feed-forward neural networks with one non-linear hidden layer and linear output units. The transfer function in the hidden layer are either bell-shaped or sigmoid. In the bell-shaped case, we show how Bernstein polynomials on one hand and the theory of the heat equation on the other are relevant for understanding the properties of the corresponding networks. In particular, these techniques yield simple proofs of universal approximation properties, i.e. of the fact that any reasonable function can be approximated to any degree of precision by a linear combination of bell-shaped functions. In addition, in this framework the problem of learning is equivalent to the problem of reversing the time course of a diffusion process. The results obtained in the bell-shaped case can then be applied to the case of sigmoid transfer functions in the hidden layer, yielding similar universality results. A conjecture related to the problem of generalization is briefly examined.

## 1  INTRODUCTION

Bell-shaped response curves are commonly found in biological neurons whenever a natural metric exist on the corresponding relevant stimulus variable (orientation, position in space, frequency, time delay, ...). As a result, they are often used in neural models in different context ranging from resolution enhancement and interpolation to learning (see, for instance, Baldi et al. (1988), Moody et al. (1989)

and Poggio et al. (1990)). Consider then the problem of approximating a function $y = f(x)$ by a weighted sum of bell-shaped functions $B(k,x)$, i. e. of finding a suitably good set of weights $H(k)$ satisfying

$$f(x) \approx \sum_k H(k)B(k,x). \tag{1}$$

In neural network terminology, this corresponds to using a feed-forward network with a unique hidden layer of bell-shaped units and a linear ouput layer. In this note, we first briefly point out how this question is related to two different mathematical concepts: Bernstein Polynomials on one hand and the Heat Equation on the other. The former shows how such an approximation is always possible for any reasonable function whereas through the latter the problem of learning, that is of finding $H(k)$, is equivalent to reversing the time course of a diffusion process. For simplicity, the relevant ideas are presented in one dimension. However, the extension to the general setting is straightforward and will be sketched in each case. We then indicate how these ideas can be applied to similar neural networks with sigmoid transfer functions in the hidden layer. A conjecture related to the problem of generalization is briefly examined.

## 2    BERNSTEIN POLYNOMIALS

In this section, without any loss of generality, we assume that all the functions to be considered are defined over the interval [0,1]. For a fixed integer $n$, there are $n$ Bernstein polynomials of degree $n$ (see, for instance, Feller (1971)) given by

$$B_n(k,x) = \binom{n}{k} x^k (1-x)^{n-k}. \tag{2}$$

$B_n(k,x)$ can be interpreted as being the probability of having $k$ successes in a coin flipping experiment of duration $n$, where $x$ represents the probability of a single success. It is easy to see that $B_n(k,x)$ is bell-shaped and reaches its maximum for $x = k/n$. Can we then approximate a function $f$ using linear combinations of Bernstein polynomials of degree $n$? Let us first consider, as an example, the simple case of the identity function $f(x) = x$ ($x \in [0,1]$). If we interpret $x$ as the probability of success on a single coin toss, then the expected number of successes in $n$ trials is given by

$$nx = \sum_{k=0}^{n} k \binom{n}{k} x^k (1-x)^{n-k} \tag{3}$$

or equivalently

$$f(x) = \sum_{k=0}^{n} f(\frac{k}{n}) \binom{n}{k} x^k (1-x)^{n-k}. \tag{4}$$

The remarkable theorem of Bernstein is that (4) remains approximately true for a general function $f$. More precisely:

**Theorem:** *Assume $f$ is a bounded function defined over the interval $[0,1]$. Then*

$$\lim_{n \to \infty} \sum_{k=0}^{n} f(\frac{k}{n}) \binom{n}{k} x^k (1-x)^{n-k} = f(x) \tag{5}$$

*at any point $x$ where $f$ is continuous. Moreover, if $f$ is continuous everywhere, the sequence in (5) approaches $f$ uniformly.*

**Proof:** The proof is beautiful and elementary. It is easy to see that

$$|f(x) - \sum_{k=0}^{n} f(\frac{k}{n})\binom{n}{k}x^k(1-x)^{n-k}| \leq \sum_{|x-\frac{k}{n}|<\delta} |f(x) - f(\frac{k}{n})|\binom{n}{k}x^k(1-x)^{n-k}$$

$$+ \sum_{|x-\frac{k}{n}|\geq\delta} |f(x) - f(\frac{k}{n})|\binom{n}{k}x^k(1-x)^{n-k}$$

for any $0 \leq \delta \leq 1$. To bound the first term in the right hand side of this inequality, we use the fact that for fixed $\epsilon$ and for $n$ large enough, at a point of continuity $x$, we can find a $\delta$ such that $|f(x) - f(\frac{k}{n})| < \epsilon$ as soon as $|x - \frac{k}{n}| < \delta$. Thus the first term is bounded by $\epsilon$. If $f$ is continuous everywhere, then it is uniformly continuous and $\delta$ can be found independently of $x$. For the second term, since $f$ is bounded $(|f(x)| \leq M)$, we have $|f(x) - f(\frac{k}{n})| \leq 2M$. Now we use Tchebycheff inequality $(P(|X - E(X)| \geq a) \leq (\text{Var}X)/a^2)$ to bound the tail of the binomial series

$$|\sum_{|x-\frac{k}{n}|\geq\delta} \binom{n}{k}x^k(1-x)^{n-k}| \leq \frac{nx(1-x)}{\delta^2 n^2} \leq \frac{1}{4n\delta^2}.$$

Collecting terms, we finally get

$$|f(x) - \sum_{k=0}^{n} f(\frac{k}{n})\binom{n}{k}x^k(1-x)^{n-k}| \leq \epsilon + \frac{M}{2n\delta^2}.$$

which completes the proof.

Bernsteins's theorem provides a probabilistic constructive proof of Weierstrass theorem which asserts that every continuous function over a compact set can be uniformly approximated by a sequence of polynomials. Its "connectionist" interpretation is that every reasonable function can be computed by a two layer network consisting of one array of equally spaced bell-shaped detectors feeding into one linear output unit. In addition, the weighting function $H(k)$ is the function $f$ itself (see also Baldi et al. (1988)). Notice that the shape of the functions $B_n(k, x)$ in the array depends on $k$: in the center $(k \approx n/2)$ they are very symmetric and similar to gaussians, as one moves towards the periphery the shape becomes less symmetric. Two additional significant properties of Bernstein polynomials are that, for fixed $n$, they form a partition of unity: $\sum_k B_n(k, x) = (x + (1-x))^n = 1$ and that they have constant energy $\int_0^1 B_n(k, x) = 1/(n + 1)$. One important advantage of the approximation defined by (5) is its great smoothness. If $f$ is differentiable, then not only (5) holds but also

$$\lim_{n\to\infty} \frac{d}{dx}(\sum_{k=0}^{n} f(\frac{k}{n})\binom{n}{k}x^k(1-x)^{n-k}) \to \frac{df}{dx} \qquad (6)$$

uniformly on $[0,1]$ and the same is true for higher order derivatives (see, for instance, Davis (1963)). Thus the Bernstein polynomials provide simultaneous approximation of a function and of its derivatives. In particular, they preserve the convexity properties of the function $f$ being approximated and mimic extremely well its qualitative behavior. The price to be paid is in precision, for the convergence in (5) can sometimes be slow. Good qualitative properties of the approximation may be relevant for biological systems, whereas precision there may not be a problem, especially in light of the fact that $n$ is often large.

Finally, this approach can be extended to the general case of an input space with $d$ dimensions by defining the generalized Bernstein polynomials

$$B_{n_1,...,n_d}(k_1,...,k_d,x_1,...,x_d) = \binom{n_1}{k_1}...\binom{n_d}{k_d}x_1^{k_1}(1-x_1)^{n_1-k_1}...x_d^{k_d}(1-x_d)^{n_d-k_d}. \tag{7}$$

If $f(x_1,...,x_d)$ is a continuous function over the hypercube $[0,1]^d$, then

$$\sum_{k_1=0}^{n_1}...\sum_{k_d=0}^{n_d} f(\frac{k_1}{n_1},...,\frac{k_d}{n_d})B_{n_1,...,n_d}(k_1,...,k_d,x_1,...,x_d) \tag{8}$$

approaches uniformly $f$ on $[0,1]^d$ as $\min n_i \to \infty$.

## 3   LEARNING AND THE HEAT EQUATION

Consider again the general problem of approximating a function $f$ by a linear combination of bell-shaped functions, but where now the bell-shaped functions are gaussians $B(w,x)$, of the form

$$B(w,x) = \frac{1}{\sqrt{2\pi}\sigma}e^{-(x-w)^2/2\sigma^2}. \tag{9}$$

The fixed centers $w$ of the gaussians are distributed in space according to a density $\mu(w)$ (this enables one to treat the continuous and discrete case together and also to include the case where the centers are not evenly distributed). This idea was directly suggested by a presentation of R. Durbin (1990), where the limiting case of an infinite number of logistic hidden units in a connectionist network was considered. In this setting, we are trying to express $f$ as

$$f(x) \approx \int_{-\infty}^{+\infty} h(w)\frac{1}{\sqrt{2\pi}\sigma}e^{-(x-w)^2/2\sigma^2}\mu(w)dw \tag{10}$$

or

$$f(x) \approx \int_{-\infty}^{+\infty} \frac{1}{\sqrt{2\pi}\sigma}H(w)e^{-(x-w)^2/2\sigma^2}dw \tag{11}$$

where $H = h\mu$. Now, diffusion processes or propagation of heat are usually modeled by a partial differential equation of the type

$$\frac{\partial u}{\partial t} = \frac{\partial^2 u}{\partial x^2} \tag{12}$$

(the heat equation) where $u(x,t)$ represents the temperature (or the concentration) at position $x$ at time $t$. Given a set of initial conditions of the form $u(x,0) = g(x)$, then the distribution of temperatures at time $t$ is given by

$$u(x,t) = \int_{-\infty}^{+\infty} g(w)\frac{1}{\sqrt{4\pi t}}e^{-(x-w)^2/4t}dw. \tag{13}$$

Technically, (13) can be shown to give the correct distribution of temperatures at time $t$ provided $g$ is continuous, $|g(x)| = O(\exp(hx^2))$ and $0 \leq t < 1/4h$. Under these conditions, it can be seen that $u(x,t) = O(\exp(kx^2))$ for some constant $k > 0$ (depending on $h$) and is the unique solution satisfying this property (see Friedman (1964) and John (1975) for more details).

The connection to our problem now becomes obvious. If the initial set of temperatures is equal to the weights in the network ($H(w) = g(w)$), then the function computed by the network is equal to the temperature at $x$ at time $t = \sigma^2/2$. Given a function $f(x)$ we can view it as a description of temperature values at time $\sigma^2/2$; *the problem of learning, i. e. of determining the optimal $h(w)$ (or $H(w)$) consists in finding a distribution of initial temperatures at time $t = 0$ from which $f$ could have evolved.* In this sense, learning is equivalent to reversing time in a diffusion process. If the continuous case is viewed as a limiting case where units with bell-shaped tuning curves are very densely packed, then it is reasonable to consider that, as the density is increased, the width $\sigma$ of the curves tends to 0. As $\sigma \rightarrow 0$, the final distribution of temperatures approaches the initial one and this is another heuristic way of seeing why the weighting function $H(w)$ is identical to the function being learnt.

In the course of a diffusion or heat propagation process, the integral of the concentration (or of the temperature) remains constant in time. Thus the temperature distribution is similar to a probability distribution and we can define its entropy

$$E(u(x,t)) = -\int_{-\infty}^{+\infty} u(x,t)\ln u(x,t)dx. \tag{14}$$

It is easy to see that the heat equation tends to increase $E$. Therefore learning can also be viewed as a process by which $E$ is minimized (within certain time boundaries constraints). This is intuitively clear if we think of learning as an attempt to evolve an initially random distribution of connection weights and concentrate it in one or a few restricted regions of space.

In general, the problem of solving the heat equation backwards in time is difficult: physically it is an irreversible process and mathematically the problem is ill-posed in the sense of Hadamard. The solution does not always exist (for instance, the final set of temperatures must be an analytic function), or exists only over a limited period of time and, most of all, small changes in the final set of temperatures can lead to large changes in the initial set of temperatures) (see, for instance, John (1955)). However, the problem becomes well-posed if the final set of temperatures has a compact Fourier spectrum (see Miranker (1961); alternatively, one could use a regularization approach as in Franklin (1974)). In a connectionist framework, one usually seeks a least square approximation to a given function. The corresponding error functional is convex (the heat equation is linear) and therefore a solution always exists. In addition, the problem is usually not ill-posed because the functions

to be learnt have a bounded spectrum and are often known only through a finite sample. Thus learning from examples in networks consisting of one hidden layer of gaussians units and a linear output unit is relatively straightforward, for the landscape of the usual error function has no local minima and the optimal set of weights can be found by gradient descent or directly, essentially by linear regression. To be more precise, we can write the error function in the most general case in the form:

$$E(h(w)) = \int [f(x) - \int h(u)e^{-(x-u)^2/2\sigma^2}\mu(u)du]^2\nu(x)dx \qquad (15)$$

where $\mu$ and $\nu$ are the measures defined on the weights and the examples respectively. The gradient, as in the usual back-propagation of errors, is given by:

$$\frac{\partial E}{\partial h(w)} = -2\int [f(x) - \int H(u)e^{-(x-u)^2/2\sigma^2}du]e^{-(x-w)^2/2\sigma^2}\mu(w)\nu(x)dx. \qquad (16)$$

Thus the critical weights of (15) where $\mu(w) \neq 0$ are characterized by the relation

$$\int f(x)e^{-(x-w)^2/2\sigma^2}\nu(x)dx = \int\int H(u)e^{-(x-w)^2/2\sigma^2}e^{-(x-u)^2/2\sigma^2}\nu(x)dudx. \qquad (17)$$

If now we assume that the centers of the gaussians in the hidden layer occupy a (finite or infinite) set of isolated points $w_i$, (17) can be rewritten in matrix form as

$$B = AH(u) \qquad (18)$$

where $B_i = \int f(x)\exp(-(x-w_i)^2/2\sigma^2)\nu(x)dx$, $H(u)_i = h(u_i)\mu(u_i)$ and $A$ is the real symmetric matrix with entries

$$A_{ij} = \int e^{-(x-w_i)^2/2\sigma^2}e^{-(x-u_j)^2/2\sigma^2}\nu(x)dx. \qquad (19)$$

Usually $A$ is invertible, so that $H(u) = A^{-1}B$ which, in turn, yields $h(u_i) = H(u_i)/\mu(u_i)$.

Finally, everything can be extended without any difficulty to $d$ dimensions, where the typical solution of $\nabla^2 u = \partial u/\partial t$ is given by

$$u(x_1, ..., x_d, t) = \int_{-\infty}^{+\infty}...\int_{-\infty}^{+\infty} g(w)\frac{1}{(4\pi t)^{d/2}}e^{-\sum_i (x_i-w_i)^2/4t}dw_1...dw_d \qquad (20)$$

with, under some smoothness assumptions, $u(x,t) \to g(x)$ as $t \to 0$.

## Remark

For an application to a discrete setting consider, as in Baldi et al. (1988), the sum

$$l(x) = \sum_{k=-\infty}^{+\infty} \frac{k}{\sqrt{2\pi}\sigma}e^{-(x-k)^2/2\sigma^2}.$$

For an initial gaussian distribution of temperatures $u(x,0)$ of the form $(1/\sqrt{2\pi})\exp(-x^2/2\eta^2)$, the distribution $u(x,t)$ of temperatures at time $t$ is also gaussian, centered at the origin, but with a larger standard deviation which, using (13), is given by $(\eta^2+2t)^{1/2}$. Thus, if we imagine that at time 0 a temperature equal

to $k$ has been injected (with a very small $\eta$) at each integer location along the real axis, then $l(x)$ represents the distribution of temperatures at time $t = (\sigma^2 - \eta^2)/2$. Intuitively, it is clear that as $\sigma$ is increased (i.e. as we wait longer) the distribution of temperatures becomes more and more linear.

(2) It is aesthetically pleasing that the theory of the heat equation can also be used to give a proof of Weierstrass theorem. For this purpose, it is sufficient to observe that, for a given continuous function $g$ defined over a closed interval $[a, b]$, the function $u(x, t)$ given by (13) is an analytic function in $x$ at a fixed time $t$. By letting $t \to 0$ and truncating the corresponding series, one can get polynomial approximations converging uniformly to $g$.

# 4   THE SIGMOID CASE

We now consider the case of a neural network with one hidden layer of sigmoids and one linear output unit. The output of the network can be written as a transform

$$out(x) = \int \sigma(w.x) h(w) \mu(w) dw \tag{21}$$

where $x$ is the input vector and $w$ is a weight vector which is characteristic of each hidden unit (i. e. each hidden units is characterized by the vector of weights on its incoming input lines rather than, for instance, its spatial location). Assume that the inputs and the weights are normalized, i.e. $||x|| = ||w|| = 1$ and that the weight vectors cover the n-dimensional sphere uniformly (or, in the limit, that there is a vector for each point on the sphere). Then for a given input $x$, the scalar products $w.x$ are maximal and close to 1 in the region of the sphere corresponding to hidden units where $w$ and $x$ are colinear and decay as we move away till they reach negative values close to $-1$ in the antipodal region. When these scalar products are passed through an appropriate sigmoid, a bell-shaped pattern of activity is created on the surface of the sphere and from then on we are reduced to the previous case. Thus the previous results can be extended and in particular we have a heuristic simple proof that the corresponding networks have universal approximation properties (see, for instance, Hornik et al. (1989)). Notice that intuitively the reason is simple for we end up we something like a grand-mother cell per pattern or cluster of patterns.

If we assume that initially $\mu(w) \neq 0$ everywhere, then it is clear that for learning via LMS optimization we can take $\mu$ to be fixed and adjust only the output weights $h$. But the problem then is convex and without local minima. This suggests that in the limit of an extremely large number of hidden units, the landscape of the error function is devoid of local minima and learning becomes very smooth. This result is consistent with the conjecture that under reasonable assumptions, as we progressively increase the number of hidden units, learning goes from being impossible, to being possible but difficult and lengthy, to being relatively easy and quick to trivial. And if so what is the nature of these transitions? This picture is also consistent with certain simulation results reported by several authors, whereby optimal performance and generalization is not best obtained by training for a very long time a minimal size highly constrained network, but rather by training for a shorter time (until the validation error begins to go up (see Baldi and Chauvin (1991))) a larger network with extra hidden units.

## Acknowledgements

This work is supported by NSF grant DMS-8914302 and ONR contract NAS7-100/918. We would like to thank Y. Rinott for useful discussions.

## Footnotes

*and Division of Biology, California Institute of Technology. The complete title of this paper should read: "Computing with arrays of bell-shaped and sigmoid functions. Bernstein polynomials, the heat equation and universal approximation properties".

## References

Baldi, P. and Heiligenberg, W. (1988) How sensory maps could enhance resolution through ordered arrangements of broadly tuned receivers. Biological Cybernetics, **59**, 313-318.

Baldi, P. and Chauvin, Y. (1991) A study of generalization in simple networks. Submitted for publication.

Davis, P. J. (1963) Interpolation and approximation. Blaisdell.

Durbin, R. (1990) Presented at the Neural Networks for Computing Conference, Snowbird, Utah.

Feller, W. (1971) An introduction to probability theory and its applications. John Wiley & Sons

Franklin, J. N. (1974) On Tikhonov's method for ill-posed problems. Mathematics of Computation, 28, 128, 889-907.

Friedman, A. (1964) Partial differential equations of parabolic type. Prentice-Hall.

Hornik, K., Stinchcombe, M. and White, H. (1989) Multilayer feedforward networks are universal approximators. Neural Networks, **2**, 5, 359-366.

John, F. (1955) Numerical solutions of the equation of heat conduction for preceding times. Ann. Mat. Pura Appl., ser. IV, vol. 40, 129-142.

John, F. (1975) Partial differential equations. Springer Verlag.

Miranker, W. L. (1961) A well posed problem for the backward heat equation. Proceedings American Mathematical Society, **12**, 243-247.

Moody, J. and Darken, C. J. (1989) Fast learning in networks of locally-tuned processing units. Neural Computation, **1**, 2, 281-294.

Poggio, T. and Girosi, F. (1990) Regularization algorithms for learning that are equivalent to multilayer networks. Science, **247**, 978-982.